# Minimizing Sparse High-Order Energies by Submodular Vertex-Cover

**Andrew Delong**
University of Toronto
andrew.delong@gmail.com

**Olga Veksler**
Western University
olga@csd.uwo.ca

**Anton Osokin**
Moscow State University
anton.osokin@gmail.com

**Yuri Boykov**
Western University
yuri@csd.uwo.ca

## Abstract

Inference in high-order graphical models has become important in recent years. Several approaches are based, for example, on generalized message-passing, or on transformation to a pairwise model with extra 'auxiliary' variables. We focus on a special case where a much more efficient transformation is possible. Instead of adding variables, we transform the original problem into a comparatively small instance of *submodular vertex-cover*. These vertex-cover instances can then be attacked by existing algorithms (*e.g.* belief propagation, QPBO), where they often run 4–15 times faster and find better solutions than when applied to the original problem. We evaluate our approach on synthetic data, then we show applications within a fast hierarchical clustering and model-fitting framework.

## 1 Introduction

MAP inference on graphical models is a central problem in machine learning, pattern recognition, and computer vision. Several algorithms have emerged as practical tools for inference, especially for graphs containing only unary and pairwise factors. Prominent examples include belief propagation [30], more advanced message passing methods like TRW-S [21] or MPLP [33], combinatorial methods like $\alpha$-expansion [6] (for 'metric' factors) and QPBO [32] (mainly for binary problems). In terms of optimization, these algorithms are designed to minimize objective functions (energies) containing unary and pairwise terms.

Many inference problems must be modeled using high-order terms, not just pairwise, and such problems are increasingly important for many applications. Recent developments in high-order inference include, for example, high-arity CRF potentials [19, 38, 25, 31], cardinality-based potentials [13, 34], global potentials controlling the appearance of labels [24, 26, 7], learning with high-order loss functions [35], among many others.

One standard approach to high-order inference is to transform the problem to the pairwise case and then simply apply one of the aforementioned 'pairwise' algorithms. These transformations add many 'auxiliary' variables to the problem but, if the high-order terms are *sparse* in the sense suggested by Rother *et al.* [31], this can still be a very efficient approach. There can be several equivalent high-order-to-pairwise transformations, and this choice affects the difficulty of the resulting pairwise inference problem. Choosing the 'easiest' transformation is not trivial and has been explicitly studied, for example, by Gallagher *et al.* [11].

Our work is about fast energy minimization (MAP inference) for particularly sparse, high-order "pattern potentials" used in [25, 31, 29]: each energy term prefers a specific (but arbitrary) assignment to its subset of variables. Instead of directly transforming the high-order problem to pairwise, we transform the entire problem to a comparatively small instance of *submodular vertex-cover* (SVC). The vertex-cover implicitly provides a solution to the original high-order problem. The SVC instance can itself be converted to pairwise, and standard inference techniques run much faster and are often more effective on this compact representation.

We also show that our 'sparse' high-order energies naturally appear when trying to solve hierarchical clustering problems using the algorithmic approach called *fusion moves* [27], also conceptually known as *optimized crossover* [1]. Fusion is a powerful *very large-scale neighborhood search* technique [3] that in some sense generalizes $\alpha$-expansion. The fusion approach is not standard for the kind of clustering objective we will consider, but we believe it is an interesting optimization strategy.

The remainder of the paper is organized as follows. Section 2 introduces the class of high-order energies we consider, then derives the transformation to SVC and the subsequent decoding. Section 3 contains experiments that suggest significant speedups, and discusses possible applications.

## 2   Sparse High-Order Energies Reducible to SVC

In what follows we use $\boldsymbol{x}$ to denote a vector of binary variables, $\mathbf{x}_P$ to denote product $\prod_{i \in P} x_i$, and $\bar{\mathbf{x}}_Q$ to denote $\prod_{i \in Q} \bar{x}_i$. It will be convenient to adopt the convention that $\mathbf{x}_{\{\}} = 1$ and $\bar{\mathbf{x}}_{\{\}} = 1$. We always use $i$ to denote a variable index from $\mathcal{I}$, and $j$ to denote a clique index from $\mathcal{V}$.

It is well-known that any pseudo-boolean function (binary energy) can be written in the form

$$F(\boldsymbol{x}) = \sum_{i \in \mathcal{I}} a_i x_i \; - \; \sum_{j \in \mathcal{V}} b_j \mathbf{x}_{P_j} \bar{\mathbf{x}}_{Q_j} \tag{1}$$

where each clique $j$ has coefficient $-b_j$ with $b_j \geq 0$, and is defined over variables in sets $P_j, Q_j \subseteq \mathcal{I}$. Our approach will be of practical interest only when, roughly speaking, $|\mathcal{V}| \ll |\mathcal{I}|$.

For example, if $\boldsymbol{x} = (x_1, \ldots, x_7)$ then a clique $j$ with $P_j = \{2, 3\}$ and $Q_j = \{4, 5, 6\}$ will explicitly reward binary configuration $(\cdot, 1, 1, 0, 0, 0, \cdot)$ by the amount $b_j$ (depicted as $b_1$ in Figure 1). If there are several overlapping (and conflicting) cliques, then the minimization problem can be difficult.

A standard way to minimize $F(\boldsymbol{x})$ would be to substitute each $-b_j \mathbf{x}_{P_j} \bar{\mathbf{x}}_{Q_j}$ term with a collection of equivalent pairwise terms. In our experiments, we used the substitution $-\mathbf{x}_{P_j} \bar{\mathbf{x}}_{Q_j} = -1 + \min_{y \in \{0,1\}} y + \sum_{i \in P_j} \bar{x}_i \bar{y} + \sum_{i \in Q_j} x_i \bar{y}$ where $y$ is an auxiliary variable. This is like the Type-II transformation in [31], and we found that it worked better than Type-I for our experiments. However, we aim to minimize $F(\boldsymbol{x})$ in a novel way, so first we review the *submodular vertex-cover* problem.

### 2.1   Review of Submodular Vertex-Cover

The classic *minimum-weighted vertex-cover* (VC) problem can be stated as a 0-1 integer program where variable $u_j = 1$ if and only if vertex $j$ is included in the cover.

$$\text{(VC)} \quad \text{minimize} \;\; \textstyle\sum_{j \in \mathcal{V}} w_j u_j \tag{2}$$
$$\text{subject to} \;\; u_j + u_{j'} \geq 1 \quad \forall \{j, j'\} \in \mathcal{E} \tag{3}$$
$$u_j \in \{0, 1\}.$$

Without loss of generality one can assume $w_j > 0$ and $j \neq j'$ for all $\{j, j'\} \in \mathcal{E}$. If the graph $(\mathcal{V}, \mathcal{E})$ is bipartite, then we call the specialized problem VC-B and it can be solved very efficiently by specialized *bipartite maximum flow* algorithms such as [2].

A function $f(\boldsymbol{x})$ is called *submodular* if $f(\boldsymbol{x} \wedge \boldsymbol{y}) + f(\boldsymbol{x} \vee \boldsymbol{y}) \leq f(\boldsymbol{x}) + f(\boldsymbol{y})$ for all $\boldsymbol{x}, \boldsymbol{y} \in \{0, 1\}^{\mathcal{V}}$ where $(\boldsymbol{x} \wedge \boldsymbol{y})_j = x_j y_j$ and $(\boldsymbol{x} \vee \boldsymbol{y})_j = 1 - \bar{x}_j \bar{y}_j$. A submodular function can be minimized in strongly polynomial time by combinatorial methods [17], but becomes NP-hard when subject to arbitrary covering constraints like (3).

The *submodular vertex-cover* (SVC) problem generalizes VC by replacing the linear (*modular*) objective (2) with an arbitrary submodular objective,

$$\text{(SVC)} \quad \text{minimize} \;\; f(\boldsymbol{u}) \tag{4}$$
$$\text{subject to} \;\; u_j + u_{j'} \geq 1 \quad \forall \{j, j'\} \in \mathcal{E}$$
$$u_j \in \{0, 1\}.$$

Iwata & Nagano [18] recently showed that when $f(\cdot) \geq 0$ a 2-approximation can be found in polynomial time and that this is the best constant-ratio bound achievable. It turns out that a half-integral relaxation $u_j \in \{0, \frac{1}{2}, 1\}$ (call this problem SVC-H), followed by upward rounding, gives

a 2-approximation much like for standard VC. They also show how to transform any SVC-H instance into a *bipartite* instance of SVC (see below); this extends a classic result by Nemhauser & Trotter [28], allowing specialized combinatorial algorithms like [17] to solve the relaxation.

In the *bipartite submodular vertex-cover* (SVC-B) problem, the graph nodes $\mathcal{V}$ can be partitioned into sets $\mathcal{J}, \mathcal{K}$ so the binary variables are $\boldsymbol{u} \in \{0,1\}^{\mathcal{J}}, \boldsymbol{v} \in \{0,1\}^{\mathcal{K}}$ and we solve

$$\text{(SVC-B)} \quad \text{minimize } f(\boldsymbol{u}) + g(\boldsymbol{v}) \tag{5}$$
$$\text{subject to } u_j + v_k \geq 1 \quad \forall \{j,k\} \in \mathcal{E}$$
$$u_j, v_k \in \{0,1\} \, \forall j \in \mathcal{J}, k \in \mathcal{K}$$

where both $f(\cdot)$ and $g(\cdot)$ are submodular functions. This SVC-B formulation is a trivial extension of the construction in [18] (they assume $g = f$), and their proof of tractability extends easily to (5).

## 2.2 Solving Bipartite SVC with Min-Cut

It will be useful to note that if $f$ and $g$ above can be written in a special manner, SVC-B can be solved by fast $s$-$t$ minimum cut instead of by [17, 15]. Suppose we have an SVC-B instance $(\mathcal{J}, \mathcal{K}, \mathcal{E}, f, g)$ where we can write submodular $f$ and $g$ as

$$f(\boldsymbol{u}) = \sum_{S \in \mathcal{S}^0} w_S \mathbf{u}_S, \text{ and } g(\boldsymbol{v}) = \sum_{S \in \mathcal{S}^1} w_S \mathbf{v}_S. \tag{6}$$

Here $\mathcal{S}^0$ and $\mathcal{S}^1$ are collections of subsets of $\mathcal{J}$ and $\mathcal{K}$ respectively, and typescript $\mathbf{u}_S$ denotes product $\prod_{j \in S} u_j$ throughout (as distinct from typescript $\boldsymbol{u}$, which denotes a vector).

**Proposition 1.** *If $w_S \leq 0$ for all $|S| \geq 2$ in (6), then* SVC-B *reduces to $s$-$t$ minimum cut.*

*Proof.* We can define an equivalent problem over variables $u_j$ and $z_k = \bar{v}_k$. With this substitution, the covering constraints become $u_j \geq z_k$. Since "$g(\boldsymbol{v})$ submodular in $\boldsymbol{v}$" implies "$g(\mathbf{1}-\boldsymbol{v})$ submodular in $\boldsymbol{v}$," letting $\bar{g}(\boldsymbol{z}) = g(\bar{\boldsymbol{z}}) = g(\boldsymbol{v})$ means $\bar{g}(\boldsymbol{z})$ is submodular as a function of $\boldsymbol{z}$. Minimizing $f(\boldsymbol{u}) + \bar{g}(\boldsymbol{z})$ subject to $u_j \geq z_k$ is equivalent to our original problem. Since $u_j \geq z_k$ can be enforced by large (submodular) penalty on assignment $\bar{u}_j z_k$, SVC-B is equivalent to

$$\text{minimize } f(\boldsymbol{u}) + \bar{g}(\boldsymbol{z}) + \sum_{(j,k) \in \mathcal{E}} \eta \bar{u}_j z_k \quad \text{where } \eta = \infty. \tag{7}$$

When $f$ and $g$ take the form (6), we have $\bar{g}(\boldsymbol{z}) = \sum_{S \in \mathcal{S}^1} w_S \bar{\mathbf{z}}_S$ where $\bar{\mathbf{z}}_S$ denotes product $\prod_{k \in S} \bar{z}_k$.

If $w_S \leq 0$ for all $|S| \geq 2$, we can build an $s$-$t$ minimum cut graph corresponding to (7) by directly applying the constructions in [23, 10]. We can do this because each term has coefficient $w_S \leq 0$ when written as $u_1 \cdots u_{|S|}$ or $\bar{z}_1 \cdots \bar{z}_{|S|}$, *i.e.* either all complemented or all uncomplemented. $\square$

## 2.3 Transforming $F(\boldsymbol{x})$ to SVC

To get a sense for how our transformation works, see Figure 1. The transformation is reminiscent of the binary *dual* of a Constraint Satisfaction Problem (CSP) [37]. The vertex-cover construction of [4] is actually a special *linear* (modular) case of our transformation (details in Proposition 2).

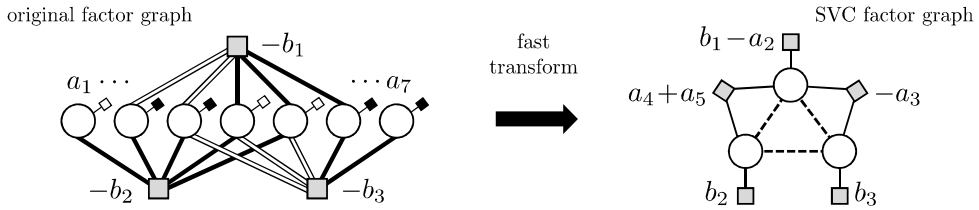

Figure 1: Left: factor graph $F(\boldsymbol{x}) = \sum_{i=1}^{7} a_i x_i - b_1 x_2 x_3 \bar{x}_4 \bar{x}_5 \bar{x}_6 - b_2 \bar{x}_1 \bar{x}_2 \bar{x}_3 \bar{x}_4 \bar{x}_5 - b_3 x_3 x_4 x_5 \bar{x}_6 \bar{x}_7$. A small white square indicates $a_i > 0$, a black square $a_i < 0$. A hollow edge connecting $x_i$ to factor $j$ indicates $i \in P_j$, and a filled-in edge indicates $i \in Q_j$. Right: factor graph of our corresponding SVC instance. High-order factors of the original problem, shown with gray squares on the left, are transformed into variables of SVC problem. Covering constraints are shown as dashed lines. Two pairwise factors are formed with coefficients $w_{\{1,3\}} = -a_3$ and $w_{\{1,2\}} = a_4 + a_5$, both $\leq 0$.

**Theorem 1.** *For any $F(\boldsymbol{x})$ there exists an instance of* SVC *such that an optimum $\boldsymbol{x}^* \in \{0,1\}^{\mathcal{I}}$ for $F$ can be computed from an optimal vertex-cover $\boldsymbol{u}^* \in \{0,1\}^{\mathcal{V}}$.*

*Proof.* First we give the construction for SVC instance $(\mathcal{V}, \mathcal{E}, f)$. Introduce auxiliary binary variables $\boldsymbol{u} \in \{0,1\}^{\mathcal{V}}$ where $\bar{u}_j = \mathbf{x}_{P_j} \bar{\mathbf{x}}_{Q_j}$. Because each $b_j \geq 0$, minimizing $F(\boldsymbol{x})$ is equivalent to the 0-1 integer program with non-linear constraints

$$\text{minimize } F(\boldsymbol{x}, \boldsymbol{u})$$
$$\text{subject to } \bar{u}_j \leq \mathbf{x}_{P_j} \bar{\mathbf{x}}_{Q_j} \quad \forall j \in \mathcal{V}. \tag{8}$$

Inequality (8) is sufficient if $b_j \geq 0$ because, for any fixed $\boldsymbol{x}$, equality $\bar{u}_j = \mathbf{x}_{P_j} \bar{\mathbf{x}}_{Q_j}$ holds for some $\boldsymbol{u}$ that minimizes $F(\boldsymbol{x}, \boldsymbol{u})$.

We try to formulate a minimization problem solely over $\boldsymbol{u}$. As a consequence of (8) we have $u_j = 0 \implies \boldsymbol{x}_{P_j} = \mathbf{1}, \boldsymbol{x}_{Q_j} = \mathbf{0}$. (We use typescript $\boldsymbol{x}_S$ to denote vector $(x_i)_{i \in S}$, whereas $\mathbf{x}_S$ denotes a product—a scalar value.) Notice that, when some $P_j$ and $Q_{j'}$ overlap, not all $\boldsymbol{u} \in \{0,1\}^{\mathcal{V}}$ can be feasible with respect to assignments $\boldsymbol{x} \in \{0,1\}^{\mathcal{I}}$. For each $i \in \mathcal{I}$, let us collect the cliques that $i$ participates in: define sets $J_i, K_i \subseteq \mathcal{V}$ where $J_i = \{j \mid i \in P_j\}$ and $K_i = \{j \mid i \in Q_j\}$. We show that $\boldsymbol{u}$ can be feasible if and only if $\mathbf{u}_{J_i} + \mathbf{u}_{K_i} \geq 1$ for all $i \in \mathcal{I}$, where $\mathbf{u}_S$ denotes a product. In other words, $\boldsymbol{u}$ can be feasible if and only if, for each $i$,

$$\begin{aligned} \exists \, u_j = 0, \ j \in J_i &\implies u_k = 1 \ \forall j \in K_i \\ \exists \, u_k = 0, \ k \in K_i &\implies u_j = 1 \ \forall j \in J_i. \end{aligned} \tag{9}$$

($\Rightarrow$) If $\bar{u}_j \leq \mathbf{x}_{P_j} \bar{\mathbf{x}}_{Q_j}$ for all $j \in \mathcal{V}$, then having $\mathbf{u}_{J_i} + \mathbf{u}_{K_i} \geq 1$ is necessary: if both $\mathbf{u}_{J_i} = 0$ and $\mathbf{u}_{K_i} = 0$ for any $i$ it would mean there exists $j \in J_i$ and $k \in K_i$ for which $\boldsymbol{x}_{P_j} = \mathbf{1}$ and $\boldsymbol{x}_{Q_k} = \mathbf{0}$, contradicting any unique assignment to $x_i$.

($\Leftarrow$) If $\mathbf{u}_{J_i} + \mathbf{u}_{K_i} \geq 1$ for all $i \in \mathcal{I}$, then we can always choose some $\boldsymbol{x} \in \{0,1\}^{\mathcal{I}}$ for which every $\bar{u}_j \leq \mathbf{x}_{P_j} \bar{\mathbf{x}}_{Q_j}$. It will be convenient to choose a *minimum cost* assignment for each $x_i$, subject to the constraints $\mathbf{u}_{J_i} = 0 \implies x_i = 1$ and $\mathbf{u}_{K_i} = 0 \implies x_i = 0$. If both $\mathbf{u}_{J_i} = \mathbf{u}_{K_i} = 1$ then $x_i$ could be either 0 or 1 so choose the best, giving

$$\boldsymbol{x}(\boldsymbol{u})_i = \begin{cases} 0 & \text{if } \mathbf{u}_{K_i} = 0 \\ 1 & \text{if } \mathbf{u}_{J_i} = 0 \\ [a_i < 0] & \text{otherwise.} \end{cases} \tag{10}$$

The assignment $\boldsymbol{x}(\boldsymbol{u})$ is feasible with respect to (8) because for any $\bar{u}_j = 1$ we have $\boldsymbol{x}(\boldsymbol{u})_{P_j} = \mathbf{1}$ and $\boldsymbol{x}(\boldsymbol{u})_{Q_j} = \mathbf{0}$.

We have completed the proof that $\boldsymbol{u}$ can be feasible if and only if $\mathbf{u}_{J_i} + \mathbf{u}_{K_i} \geq 1$. To express minimization of $F$ solely in terms of $\boldsymbol{u}$, first write (10) in equivalent form

$$\boldsymbol{x}(\boldsymbol{u})_i = \begin{cases} \mathbf{u}_{K_i} & \text{if } a_i < 0 \\ 1 - \mathbf{u}_{J_i} & \text{otherwise.} \end{cases} \tag{11}$$

Again, this definition of $\boldsymbol{x}(\boldsymbol{u})$ minimizes $F(\boldsymbol{x}, \boldsymbol{u})$ over all $\boldsymbol{x}$ satisfying inequality (8). Use (11) to write new SVC objective $f(\boldsymbol{u}) = F(\boldsymbol{x}(\boldsymbol{u}), \boldsymbol{u})$, which becomes

$$\begin{aligned} f(\boldsymbol{u}) &= \sum_{i \,:\, a_i > 0} a_i (1 - \mathbf{u}_{J_i}) \ + \sum_{i \,:\, a_i < 0} a_i \mathbf{u}_{K_i} \ - \sum_{j \in \mathcal{V}} b_j (1 - u_j) \\ &= \sum_{i \,:\, a_i > 0} -a_i \mathbf{u}_{J_i} \ + \sum_{i \,:\, a_i < 0} a_i \mathbf{u}_{K_i} \ + \sum_{j \in \mathcal{V}} b_j u_j \ + \text{const.} \end{aligned} \tag{12}$$

To collect coefficients in the first two summands of (12), we must group them by each unique clique that appears. We define set $\mathcal{S} = \{S \subseteq \mathcal{V} \mid (\exists J_i = S) \vee (\exists K_i = S)\}$ and write

$$f(\boldsymbol{u}) = \sum_{S \in \mathcal{S}} w_S \mathbf{u}_S \ + \text{const} \tag{13}$$

$$\text{where } w_S = \sum_{\substack{i \,:\, a_i > 0, \\ J_i = S}} -a_i \ + \sum_{\substack{i \,:\, a_i < 0, \\ K_i = S}} a_i \ \left( + \ b_j \text{ if } S = \{j\} \right). \tag{14}$$

Since the high-order terms $\mathbf{u}_S$ in (13) have non-positive coefficients $w_S \leq 0$, then $f(\boldsymbol{u})$ is *submodular* [5]. Also note that for each $i$ at most one of $J_i$ or $K_i$ contributes to the sum, so there are at most $|\mathcal{S}| \leq |\mathcal{I}|$ unique terms $\mathbf{u}_S$ with $w_S \neq 0$. If $|\mathcal{S}|, |\mathcal{V}| \ll |\mathcal{I}|$ then our SVC instance will be small.

Finally, to ensure (9) holds we add a covering constraint $u_j + u_k \geq 1$ whenever there exists $i$ such that $j \in J_i, k \in K_i$. For this SVC instance, an optimal covering $\boldsymbol{u}$ minimizes $F(\boldsymbol{x}(\boldsymbol{u}), \boldsymbol{u})$. $\qquad \square$

The construction in Theorem 1 suggests the entire minimization procedure below.

---

MINIMIZE-BY-SVC($F$) where $F$ is a pseudo-boolean function in the form of (1)

---

1   $w_{\{j\}} := b_j \;\; \forall j \in \mathcal{V}$
2   **for** $i \in \mathcal{I}$ **do**
3     **if**     $a_i > 0$ **then** $w_{J_i} := w_{J_i} - a_i$      (distribute $a_i$ to high-order SVC coefficients)
4     **else if** $a_i < 0$ **then** $w_{K_i} := w_{K_i} + a_i$     (where index sets $J_i$ and $K_i$ defined in Theorem 1)
5     $\mathcal{E} := \mathcal{E} \cup \{\{j,k\}\} \;\; \forall j \in J_i, k \in K_i$     (add covering constraints to enforce $\mathbf{u}_{J_i} + \mathbf{u}_{K_i} \geq 1$)
6   let $f(\boldsymbol{u}) = \sum_{S \in \mathcal{S}} w_S \mathbf{u}_S$     (define SVC objective over clique indices $\mathcal{V}$)
7   $\boldsymbol{u}^* := $ SOLVE-SVC$(\mathcal{V}, \mathcal{E}, f)$     (solve with BP, QPBO, Iwata, *etc.*)
8   **return** $\boldsymbol{x}(\boldsymbol{u}^*)$     (decode the covering as in (10))

---

One reviewer suggested an extension that scales better with the number of overlapping cliques. The idea is to formulate SVC over the elements of $\mathcal{S}$ rather than $\mathcal{V}$. Specifically, let $\boldsymbol{y} \in \{0,1\}^\mathcal{S}$ and use submodular objective $f(\boldsymbol{y}) = \sum_{S \in \mathcal{S}} w_S y_S + \sum_{j \in S}(b_j + 1) y_S \bar{y}_{\{j\}}$, where the inner sum ensures $y_S = \prod_{j \in S} y_{\{j\}}$ at a local minimum because $w_{\{j\}} \leq b_j$. For each unique pair $\{J_i, K_i\}$, add a covering constraint $y_{J_i} + y_{K_i} \geq 1$ (instead of $O(|J_i| \cdot |K_i|)$ constraints). An optimal covering $\boldsymbol{y}^*$ of $\mathcal{S}$ then gives an optimal covering of $\mathcal{V}$ by assigning $u_j = y^*_{\{j\}}$. Here we use the original construction, and still report significant speedups. See [8] for discussion of efficient implementation, and an alternate proof of Theorem 1 based on LP relaxation.

### 2.4   Special Cases of Note

**Proposition 2.** *If $\{P_j\}_{j \in \mathcal{V}}$ are disjoint and, separately, $\{Q_j\}_{j \in \mathcal{V}}$ are disjoint (equivalently each $|J_i|, |K_i| \leq 1$), then the* SVC *instance in Theorem 1 reduces to standard* VC.

*Proof.* Each $S \in \mathcal{S}$ in objective (13) must be $S = \{j\}$ for some $j \in \mathcal{V}$. The objective then becomes $f(\boldsymbol{u}) = \sum_{j \in \mathcal{V}} w_{\{j\}} u_j + \text{const}$, a form of standard VC.      □

Proposition 2 shows that the main result of [4] is a special case of our Theorem 1 when $J_i = \{j\}$ and $K_i = \{k\}$ with $j, k$ determined by two labelings being 'fused'. In Section 3, this generalization of [4] will allow us to apply a similar fusion-based algorithm to *hierarchical* clustering problems.

**Proposition 3.** *If each particular $j \in \mathcal{V}$ has either $P_j = \{\}$ or $Q_j = \{\}$, then the construction in Theorem 1 is an instance of* SVC-B. *Moreover, it is reducible to s-t minimum cut.*

*Proof.* In this case $J_i$ is disjoint with $K_{i'}$ for any $i, i' \in \mathcal{I}$, so sets $\mathcal{J} = \{j : |P_j| \geq 1\}$ and $\mathcal{K} = \{j : |Q_j| \geq 1\}$ are disjoint. Since $\mathcal{E}$ contains pairs $(j, k)$ with $j \in \mathcal{J}$ and $k \in \mathcal{K}$, graph $(\mathcal{V}, \mathcal{E})$ is bipartite. By the disjointness of any $J_i$ and $K_{i'}$, the unique clique sets $\mathcal{S}$ can be partitioned into $\mathcal{S}^0 = \{S \subseteq \mathcal{J} \mid \exists J_i = S\}$ and $\mathcal{S}^1 = \{S \subseteq \mathcal{K} \mid \exists K_i = S\}$ so that (13) can be written as in Proposition 1 and thereby reduced to *s-t* minimum cut.      □

**Corollary 1.** *If sets $\{P_j\}_{j \in \mathcal{V}}$ and $\{Q_j\}_{j \in \mathcal{V}}$ satisfy the conditions of propositions 2 and 3, then minimizing $F(\boldsymbol{x})$ reduces to an instance of* VC-B *and can be solved by bipartite maximum flow.*

We should note that even though SVC has a 2-approximation algorithm [18], this does not give us a 2-approximation for minimizing $F$ in general. Even if $F(\boldsymbol{x}) \geq 0$ for all $\boldsymbol{x}$, it does not imply $f(\boldsymbol{u}) \geq 0$ for configurations of $\boldsymbol{u}$ that violate the covering constraints, as would be required.

## 3   Applications

Even though any pseudo-boolean function can be expressed in form (1), many interesting problems would require an exponential number of terms to be expressed in that form. Only certain specific applications will naturally have $|\mathcal{V}| \ll |\mathcal{I}|$, so this is the main limitation of our approach. There may be applications in high-order segmentation. For example, when $P^n$-Potts potentials [19] are incorporated into $\alpha$-expansion, the resulting expansion step contains high-order terms that are compact in this form; in the absence of pairwise CRF terms, Proposition 3 would apply.

The $\alpha$-expansion algorithm has also been extended to optimize the *facility location* objective [7] commonly used for clustering (*e.g.* [24]). The resulting high-order terms inside the expansion step

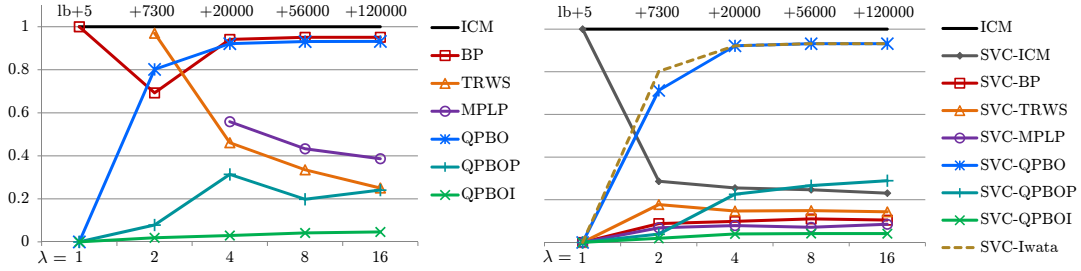

Figure 2: Effectiveness of each algorithm as strength of high-order coefficients is increased by factor of $\lambda \in \{1..16\}$. For a fixed $\lambda$, the final energy of each algorithm was normalized between 0.0 (best lower bound) and 1.0 (baseline ICM energy); the true energy gap between lower bound and baseline is indicated at top, e.g. for $\lambda = 1$ the "lb+5" means ICM was typically within 5 of the lower bound.

also take the form (1) (in fact, Corollary 1 applies here); with no need to build the 'full' high-order graph, this would allow $\alpha$-expansion to work as a fast alternative to the classic greedy algorithm for facility location, very similar to the fusion-based algorithm in [4]. However, in Section 3.2 we show that our generalized transformation allows for a novel way to optimize a *hierarchical* facility location objective. We will use a recent geometric image parsing model [36] as a specific example. First, Section 3.1 compares a number of methods on synthetic instances of energy (1).

## 3.1   Results on Synthetic Instances

Each instance is a function $F(\boldsymbol{x})$ where $\boldsymbol{x}$ represents a $100 \times 100$ grid of binary variables with random unary coefficients $a_i \in [-10, 10]$. Each instance also has $|\mathcal{J}| = 50$ high-order cliques with $b_j \in [250\lambda, 500\lambda]$ (we will vary $\lambda$), where variable sets $P_j$ and $Q_j$ each cover a random $n_j \times n_j$ and $m_j \times m_j$ region respectively (here the region size $n_j, m_j \in \{10, \ldots, 15\}$ is chosen randomly). If $P_j$ and $Q_j$ are not disjoint, then either $P_j := P_j \setminus Q_j$ or $Q_j := Q_j \setminus P_j$, as determined by a coin flip.

We tested the following algorithms: BP [30], TRW-S [21], MPLP [33], QPBO [14], and extensions QPBO-P and QPBO-I [32]. For BP we actually used the implementation provided by [21] which is very fast but, we should note, does not support message-damping; convergence of BP may be more reliable if this were supported. Algorithms were configured as follows: BP for 25 iterations (more did not help); TRW-S for 800 iterations (epsilon 1); MPLP for 2000 initial iterations + 20 clusters added + 100 iterations per tightening; QPBO-I with 5 random improve steps. We ran MPLP for a particularly long time to ensure it had ample time to tighten and converge; indeed, it always yielded the best lower bound. We also tested MINIMIZE-BY-SVC by applying each of these algorithms to solve the resulting SVC problem, and in this case also tried the Iwata-Nagano construction [18].

To transform high-order potentials to quadratic, we report results using Type-II binary reduction [31] because for TRW-S/MPLP it dominated the Type-I reduction in our experiments, and for BP and the others it made no difference. This runs counter to the conventional used of "number of supermodular terms" as an estimate of difficulty: the Type-I reduction would generate one supermodular edge per high-order term, whereas Type-II generates $|P_j|$ supermodular edges for each term ($\sum_{i \in P_j} \bar{x}_i \bar{y}$).

One minor detail is how to evaluate the 'partial' labelings returned by QPBO and QPBO-P. In the case of minimizing $F$ directly, we simply assigned such variables $x_i = [a_i < 0]$. In the case of MINIMIZE-BY-SVC we included all unlabeled nodes in the cover, which means a variable $x_i$ with $\boldsymbol{u}_{J_i}$ and $\boldsymbol{u}_{K_i}$ all unlabeled will similarly be assigned $x_i = [a_i < 0]$.

Figure 2 shows the relative performance of each algorithm, on average. When $\lambda = 1$ the high-order coefficients are relatively weak compared to the unary terms, so even ICM succeeds at finding a near-optimal energy. For larger $\lambda$ the high-order terms become more important, and we make a number of observations:

- ICM, BP, TRW-S, MPLP all perform much better when applied to the SVC problem.
- QPBO-based methods do not perform better when applied to the SVC problem.
- QPBO-I consistently gives good results; BP also gives good results if applied to SVC.
- The Iwata-Nagano construction is effectively the same as QBPO applied to SVC.

We also observed that the TRW-S lower bound was the same with or without transformation to SVC, but convergence took much fewer iterations when applied to SVC. In principle, TRW on binary problems solves the same LP relaxation as QPBO [22]. The TRW-S code finds much better solutions because it uses the final messages as hints to decode a good solution, unlike for QPBO.

Table 1 gives typical running times for each of the cases in Figure 2 on a 2.66 GHz Intel Core2 processor. Code was written in C++, but the SVC transformation was not optimized at all. Still, SVC-QBPOI is 20 times faster than QPBOI while giving similar energies on average. The overall results suggest that SVC-BP or SVC-QPBOI are the fastest ways to find a low-energy solution (bold in Table 1) on problems containing many conflicting high-order terms of the form (1). Running times were relatively consistent for all $\lambda \geq 2$.

Table 1: Typical running times of each algorithm. First row uses Type-II binary reduction on $F$, then directly runs each algorithm. Second row first transforms to SVC, does Type-II reduction, runs the algorithm, and decodes the result; times shown include all these steps.

|  | BP | TRW-S | MPLP | QPBO | QPBO-P | QPBO-I | Iwata |
|---|---|---|---|---|---|---|---|
| directly minimize $F$ | 22ms | 670ms | 25min | 30ms | 25sec | 140ms | N/A |
| MINIMIZE-BY-SVC($F$) | **5.2ms** | 19ms | 80sec | 5.4ms | 99ms | **7.2ms** | 5ms |

## 3.2  Application: Hierarchical Model-Estimation / Clustering

In clustering and multi-model estimation, it is quite common to either explicitly constrain the number of clusters, or—more relevant to our work—to penalize the number of clusters in a solution. Penalizing the number of clusters is a kind of complexity penalty on the solution. Recent examples include [24, 7, 26], but the basic idea has been used in many contexts over a long period. A classic operations research problem with the same fundamental components is *facility location*: the clients (data points) must be assigned to a nearby facility (cluster) but each facility costs money to open. This can be thought of as a labeling problem, where each data point is a variable, and there is a label for each cluster.

For hard optimization problems there is a particular algorithmic approach called *fusion* [27] or *optimized crossover* [1]. The basic idea is two take two candidate solutions (*e.g.* two attempts at clustering), and to 'fuse' the best parts of each solution, effectively stitching them together. To see this more concretely, imagine a labeling problem where we wish to minimize $E(\boldsymbol{l})$ where $\boldsymbol{l} = (l_i)_{i \in \mathcal{I}}$ is a vector of label assignments. If $\boldsymbol{l}^0$ is the first candidate labeling, and $\boldsymbol{l}^1$ is the second candidate labeling, a fusion operation seeks a binary string $\boldsymbol{x}^*$ such that the crossover labeling $\boldsymbol{l}(\boldsymbol{x}) = (l_i^{x_i})_{i \in \mathcal{I}}$ minimizes $E(\boldsymbol{l}(\boldsymbol{x}))$. In other words, $\boldsymbol{x}^*$ identifies the best possible 'stitching' of the two candidate solutions with respect to the energy.

In [4] we derived a fusion operation based on the greedy formulation of facility location, and found that the subproblem reduced to minimum-weighted vertex-cover. We will now show that the fusion operation for *hierarchical* facility location objectives requires minimizing an energy of the form (1), which we have already shown can be transformed to a *submodular* vertex-cover problem. Givoni *et al.* [12] recently proposed a message-passing scheme for hierarchical facility location, with experiments on synthetic and HIV strain data. We focus on more a computer vision-centric application: detecting a hierarchy of lines and vanishing points in images using the *geometric image parsing* objective proposed by Tretyak *et al.* [36].

The hierarchical energy proposed by [36] contains five 'layers': edges, line segments, lines, vanishing points, and horizon. Each layer provides evidence for subsequent (higher) layers, and at each level their is a complexity cost that regulates how much evidence is needed to detect a line, to detect a vanishing point, etc. For simplicity we only model edges, lines, and vanishing points, but our fusion-based framework easily extends to the full model. The purpose of our experiments are, first and foremost, to demonstrate that MINIMIZE-BY-SVC speeds up inference and, secondly, to suggest that a hierarchical clustering framework based on fusion operations (similar to non-hierarchical [4]) is an interesting and potentially worthwhile alternative to the greedy and local optimization used in state-of-the-art methods like [36].

Let $\{\boldsymbol{y}^i\}_{i \in \mathcal{I}}$ be a set of oriented edges $\boldsymbol{y}^i = (x_i, y_i, \psi_i)$ where $(x, y)$ is position in the image and $\psi$ is an angle; these bottom-level features are generated by a Canny edge detector. Let $\mathcal{L}$ be a set of candidate lines, and let $\mathcal{V}$ be a set of candidate vanishing points. These sets are built by randomly sampling: one oriented edge to generate each candidate line, and pairs of lines to generate each candidate vanishing point. Each line $j \in \mathcal{L}$ is associated with one vanishing point $k_j \in \mathcal{V}$. (If a line passes close to multiple vanishing points, a copy of the line is made for each.) We seek a labeling $\boldsymbol{l}$ where $l_i \in \mathcal{L} \cup \oslash$ identifies the line (and vanishing point) that edge $i$ belongs to, or assigns outlier label $\oslash$. Let $D_i(j) = \mathrm{dist}_j(x_i, y_i) + \mathrm{dist}_j(\psi_i)$ denote the spatial distance and angular deviation of edge $\boldsymbol{y}^i$ to line $j$, and let the outlier cost be $D_i(\oslash) = \mathrm{const}$. Similarly, let $D_j = \mathrm{dist}_j(k_j)$ be the distance of line $j$ and its associated vanishing point projected onto the Gaussian sphere (see [36]). Finally let $C_1$ and $C_{\mathrm{v}}$ denote positive constants that penalize the detection of a line and a vanishing point respectively. The hierarchical energy we minimize is

$$E(\boldsymbol{l}) = \sum_{i \in \mathcal{I}} D_i(l_i) \; + \sum_{j \in \mathcal{L}}(C_1 + D_j) \cdot [\exists l_i = j] \; + \sum_{k \in \mathcal{V}} C_{\mathrm{v}} \cdot [\exists k_{l_i} = k]. \qquad (15)$$

This energy penalizes the number of unique lines, and the number of unique vanishing points that labeling $\boldsymbol{l}$ depends on. Given two candidate labelings $\boldsymbol{l}^0, \boldsymbol{l}^1$, writing the fusion energy for (15) gives

$$E(\boldsymbol{l}(\boldsymbol{x})) = \sum_{i \in \mathcal{I}} D_i^0 + (D_i^1 - D_i^0) x_i \; + \sum_{j \in \mathcal{L}}(C_1 + D_j) \cdot (1 - \mathbf{x}_{P_j} \overline{\mathbf{x}}_{Q_j}) + \sum_{k \in \mathcal{V}} C_{\mathrm{v}} \cdot (1 - \mathbf{x}_{P_k} \overline{\mathbf{x}}_{Q_k}) \quad (16)$$

where $P_j = \{\, i \mid l_i^0 = j \,\}$, $Q_j = \{\, i \mid l_i^1 = j \,\}$, and $P_k = \{\, i \mid k_{l_i^0} = k \,\}$, $Q_k = \{\, i \mid k_{l_i^1} = k \,\}$. Notice that sets $\{P_j\}$ are disjoint with each other, but each $P_j$ is nested in subset $P_{k_j}$, so overall Proposition 2 does not apply, and so neither does the algorithm in [4].

For each image we used 10,000 edges, generated 8,000 candidate lines and 150 candidate vanishing points. We then generated 4 candidate labelings, each by allowing vanishing points to be detected in randomized order, and their associated lines to be detected in greedy order, and then we fused the labelings together by minimizing (16). Overall inference with QPBOI took 2–6 seconds per image, whereas SVC-QPBOI took 0.5-0.9 seconds per image with relative speedup of 4–6 times. The simplified model is enough to show that hierarchical clustering can be done in this new and potentially powerful way. As argued in [27], fusion is a robust approach because it *combines* the strengths—quite literally—of all methods used to generate candidates.

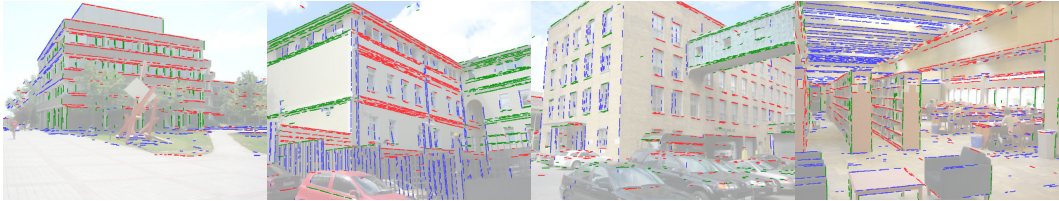

Figure 3: (Best seen in color.) Edge features color-coded by their detected vanishing point. Not shown are the detected lines that make up the intermediate layer of inference (similar to [36]). Images taken from York [9] and Eurasia [36] datasets.

**Acknowledgements**   We thank Danny Tarlow for helpful discussion regarding MPLP, and an anonymous reviewer for suggesting a more efficient way to enforce covering constraints(!). This work supported by NSERC Discovery Grant R3584A02, Canadian Foundation for Innovation (CFI), and Early Researcher Award (ERA).

# References

[1] Aggarwal, C.C., Orlin, J.B., & Tai, R.P. (1997) Optimized Crossover for the Independent Set Problem. *Operations Research* **45**(2):226–234.

[2] Ahuja, R.K., Orlin, J.B., Stein, C. & Tarjan, R.E. (1994) Improved algorithms for bipartite network flow. *SIAM Journal on Computing* **23**(5):906–933.

[3] Ahuja, R.K., Ergun, Ö., Orlin, J.B., & Punnen, A.P. (2002) A survey of very large-scale neighborhood search techniques. *Discrete Applied Mathematics* **123**(1–3):75–202.

[4] Delong, A., Veksler, O. & Boykov, Y. (2012) Fast Fusion Moves for Multi-Model Estimation. *European Conference on Computer Vision*.

[5] Boros, E. & Hammer, P.L. (2002) Pseudo-Boolean Optimization. *Discrete App. Math.* **123**(1–3):155–225.

[6] Boykov, Y., Veksler, O., & Zabih, R. (2001) Fast Approximate Energy Minimization via Graph Cuts. *IEEE Transactions on Pattern Recognition and Machine Intelligence*. **23**(11):1222–1239.

[7] Delong, A., Osokin, A., Isack, H.N., & Boykov, Y. (20120) Fast Approximate Energy Minimization with Label Costs. *International Journal of Computer Vision* **96**(1):127. Earlier version in CVPR 2010.

[8] Delong, A., Veksler, O., Osokin, A., & Boykov, Y. (2012) Minimizing Sparse High-Order Energies by Submodular Vertex-Cover. *Technical Report, Western University.*

[9] Denis, P., Elder, J., & Estrada, F. (2008) Efficient Edge-Based Methods for Estimating Manhattan Frames in Urban Imagery. *European Conference on Computer Vision.*

[10] Freedman, D. & Drineas, P. (2005) Energy minimization via graph cuts: settling what is possible. *IEEE Conference on Computer Vision and Pattern Recognition.*

[11] Gallagher, A.C., Batra, D., & Parikh, D. (2011) Inference for order reduction in Markov random fields. *IEEE Conference on Computer Vision and Pattern Recognition.*

[12] Givoni, I.E., Chung, C., & Frey, B.J. (2011) Hierarchical Affinity Propagation. *Uncertainty in AI.*

[13] Gupta, R., Diwan, A., & Sarawagi, S. (2007) Efficient inference with cardinality-based clique potentials. *International Conference on Machine Learning.*

[14] Hammer, P.L., Hansen, P., & Simeone, B. (1984) Roof duality, complementation and persistency in quadratic 0-1 optimization. *Mathematical Programming* **28**:121–155.

[15] Hochbaum, D.S. (2010) Submodular problems – approximations and algorithms. *Arxiv preprint arXiv:1010.1945.*

[16] Iwata, S., Fleischer, L. & Fujishige, S. (2001) A combinatorial, strongly polynomial-time algorithm for minimizing submodular functions. *Journal of the ACM* **48**:761–777.

[17] Iwata, S. & Orlin, J.B. (2009) A simple combinatorial algorithm for submodular function minimization. *ACM-SIAM Symposium on Discrete Algorithms.*

[18] Iwata, S. & Nagano, K. (2009) Submodular Function Minimization under Covering Constraints. *IEEE Symposium on Foundations of Computer Science.*

[19] Kohli, P., Kumar, M.P. & Torr, P.H.S. (2007) $\mathcal{P}^3$ & Beyond: Solving Energies with Higher Order Cliques. *IEEE Conference on Computer Vision and Pattern Recognition.*

[20] Kolmogorov, V. (2010) Minimizing a sum of submodular functions. *Arxiv preprint arXiv:1006.1990.*

[21] Kolmogorov, V. (2006) Convergent Tree-Reweighted Message Passing for Energy Minimization. *IEEE Transactions on Pattern Analysis and Machine Intelligence* **28**(10):1568–1583.

[22] Kolmogorov, V., & Wainwright, M.J. (2005) On the optimality of tree-reweighted max-product message-passing. *Uncertainty in Artificial Intelligence.*

[23] Kolmogorov, V. & Zabih, R. (2004) What Energy Functions Can Be Optimized via Graph Cuts? *IEEE Transactions on Pattern Analysis and Machine Intelligence* **26**(2):147–159.

[24] Komodakis, N., Paragios, N., & Tziritas, G. (2008) Clustering via LP-based Stabilities. *Neural Information Processing Systems.*

[25] Komodakis, N., & Paragios, N. (2009) Beyond pairwise energies: Efficient optimization for higher-order MRFs. *IEEE Computer Vision and Pattern Recognition.*

[26] Ladický, L., Russell, C., Kohli, P., & Torr, P.H.S (2010) Graph Cut based Inference with Co-occurrence Statistics. *European Conference on Computer Vision.*

[27] Lempitsky, V., Rother, C., Roth, S., & Blake, A. (2010) Fusion Moves for Markov Random Field Optimization. *IEEE Transactions on Pattern Analysis and Machine Inference.* **32**(9):1392–1405.

[28] Nemhauser, G.L. and Trotter, L.E. (1975) Vertex packings: Structural properties and algorithms. *Mathematical Programming* **8**(1):232–248.

[29] Osokin, A., & Vetrov, D. (2012) Submodular relaxations for MRFs with high-order potentials. *HiPot: ECCV Workshop on Higher-Order Models and Global Constraints in Computer Vision.*

[30] Pearl, J. (1988) Fusion, propagation, and structuring in belief networks. *Artificial Intell.* **29**(3):251–288.

[31] Rother, C., Kohli, P., Feng, W., & Jia, J. (2009) Minimizing sparse higher order energy functions of discrete variables. *IEEE Conference on Computer Vision and Pattern Recognition.*

[32] Rother, C., Kolmogorov, V., Lempitsky, V., & Szummer, M. (2007) Optimizing Binary MRFs via Extended Roof Duality. *IEEE Conference on Computer Vision and Pattern Recognition.*

[33] Sontag, D., Meltzer, T., Globerson, A., Jaakkola, T., & Weiss, Y. (2008) Tightening LP relaxations for MAP using message passing. *Uncertainty in Artificial Intelligence.*

[34] Tarlow, D., Givoni, I.E., & Zemel, R.S. (2010) HOPMAP: Efficient message passing with high order potentials. *International Conference on Artificial Intelligence and Statistics.*

[35] Tarlow, D., & Zemel, R. (2012) Structured Output Learning with High Order Loss Functions. *International Conference on Artificial Intelligence and Statistics.*

[36] Tretyak, E., Barinova, O., Kohli, P., & Lempitsky, V. (2011) Geometric Image Parsing in Man-Made Environments. *International Journal of Computer Vision* **97**(3):305–321.

[37] Tsang, E. (1993) Foundations of constraint satisfaction. Academic Press, London.

[38] Werner, T. (2008) High-arity Interactions, Polyhedral Relaxations, and Cutting Plane Algorithm for Soft Constraint Optimisation (MAP-MRF). *IEEE Conference on Computer Vision and Pattern Recognition.*

